# Scene Segmentation with Conditional Random Fields Learned from Partially Labeled Images

**Jakob Verbeek and Bill Triggs**

INRIA and Laboratoire Jean Kuntzmann, 655 avenue de l'Europe, 38330 Montbonnot, France

## Abstract

Conditional Random Fields (CRFs) are an effective tool for a variety of different data segmentation and labeling tasks including visual scene interpretation, which seeks to partition images into their constituent semantic-level regions and assign appropriate class labels to each region. For accurate labeling it is important to capture the global context of the image as well as local information. We introduce a CRF based scene labeling model that incorporates both local features and features aggregated over the whole image or large sections of it. Secondly, traditional CRF learning requires fully labeled datasets which can be costly and troublesome to produce. We introduce a method for learning CRFs from datasets with many unlabeled nodes by marginalizing out the unknown labels so that the log-likelihood of the known ones can be maximized by gradient ascent. Loopy Belief Propagation is used to approximate the marginals needed for the gradient and log-likelihood calculations and the Bethe free-energy approximation to the log-likelihood is monitored to control the step size. Our experimental results show that effective models can be learned from fragmentary labelings and that incorporating top-down aggregate features significantly improves the segmentations. The resulting segmentations are compared to the state-of-the-art on three different image datasets.

## 1 Introduction

In visual scene interpretation the goal is to assign image pixels to one of several semantic classes or scene elements, thus jointly performing segmentation and recognition. This is useful in a variety of applications ranging from keyword-based image retrieval (using the segmentation to automatically index images) to autonomous vehicle navigation [1].

Random field approaches are a popular way of modelling spatial regularities in images. Their applications range from low-level noise reduction [2] to high-level object or category recognition (this paper) and semi-automatic object segmentation [3]. Early work focused on generative modeling using Markov Random Fields, but recently Conditional Random Field (CRF) models [4] have become popular owing to their ability to directly predict the segmentation/labeling given the observed image and the ease with which arbitrary functions of the observed features can be incorporated into the training process. CRF models can be applied either at the pixel-level [5, 6, 7] or at the coarser level of super-pixels or patches [8, 9, 10]. In this paper we label images at the level of small patches, using CRF models that incorporate both purely local (single patch) feature functions and more global 'context capturing' feature functions that depend on aggregates of observations over the whole image or large regions.

Traditional CRF training algorithms require fully-labeled training data. In practice it is difficult and time-consuming to label every pixel in an image and most of the available image interpretation datasets contain unlabeled pixels. Working at the patch level exacerbates this problem because many patches contain several different pixel-level labels. Our CRF training algorithm handles this by allowing partial and mixed labelings and optimizing the probability for the model segmentation to be consistent with the given labeling constraints.

The rest of the paper is organized as follows: we describe our CRF model in Section 2, present our training algorithm in Section 3, provide experimental results in Section 4, and conclude in Section 5.

## 2 A Conditional Random Field using Local and Global Image Features

We represent images as rectangular grids of patches at a single scale, associating a hidden class label with each patch. Our CRF models incorporate 4-neighbor couplings between patch labels. The local image content of each patch is encoded using texture, color and position descriptors as in [10]. For texture we compute the 128-dimensional SIFT descriptor [11] of the patch and vector quantize it by nearest-neighbour assignement against a $k_s = 1000$ word texton dictionary learned by k-means clustering of all patches in the training dataset. Similarly, for color we take the 36-D hue descriptor of [12] and vector quantize it against a $k_h = 100$ word color dictionary learned from the training set. Position is encoded by overlaying the image with an $m \times m$ grid of cells ($m = 8$) and using the index of the cell in which the patch falls as its position feature. Each patch is thus coded by three binary vectors with respectively $k_s$, $k_h$ and $k_p = m^2$ bits, each with a single bit set corresponding to the observed visual word. Our CRF observation functions are simple linear functions of these three vectors. Generatively, the three modalities are modelled as being independent given the patch label.

The naive Bayes model of the image omits the 4-neighbor couplings and thus assumes that each patch label depends only on its three observation functions. Parameter estimation reduces to trivially counting observed visual word frequencies for each label class and feature type. On the MSRC 9-class image dataset this model returns an average classification rate of 67.1% (see Section 4), so isolated appearance alone does not suffice for reliable patch labeling.

In recent years models based on histograms of visual words have proven very successful for image categorization (deciding whether or not the image as a whole belongs to a given category of scenes) [13]. Motivated by this, many of our models take the global image context into account by including observation functions based on image-wide histograms of the visual words of their patches. The hope is that this will help to overcome the ambiguities that arise when patches are classified in isolation. To this end, we define a conditional model for patch labels that incorporates both local patch level features and global aggregate features. Let $x_i \in \{1, \ldots, C\}$ denote the label of patch $i$, $\mathbf{y}_i$ denote the $W$-dimensional concatenated binary indicator vector of its three visual words ($W = k_s + h_h + k_p$), and $\mathbf{h}$ denote the normalized histogram of all visual words in the image, i.e. $\sum_{\text{patches } i} \mathbf{y}_i$ normalized to sum one. The conditional probablity of the label $x_i$ is then modeled as

$$p(x_i = l | \mathbf{y_i}, \mathbf{h}) \propto \exp\left(-\sum_{w=1}^{W}(\alpha_{wl}y_{iw} + \beta_{wl}h_w)\right), \tag{1}$$

where $\alpha_{wl}, \beta_{wl}$ are $W \times C$ matrices of coefficients to be learned. We can think of this as a multiplicative combination of a local classifier based on the patch-level observation $\mathbf{y}_i$ and a global context or bias based on the image-wide histogram $\mathbf{h}$.

To account for correlations among spatially neighboring patch labels, we add couplings between the labels of neighboring patches to the single patch model (1). Let $X$ denote the collection of all patch labels in the image and $Y$ denote the collected patch features. Then our CRF model for the coupled patch labels is:

$$p(X|Y) \propto \exp\left(-E(X|Y)\right), \tag{2}$$

$$E(X|Y) = \sum_i \sum_{w=1}^{W}(\alpha_{wx_i}y_{iw} + \beta_{wx_i}h_w) + \sum_{i \sim j} \phi_{ij}(x_i, x_j), \tag{3}$$

where $i \sim j$ denotes the set of all adjacent (4-neighbor) pairs of patches $i, j$. We can write $E(X|Y)$ without explicitly including $\mathbf{h}$ as an argument because $\mathbf{h}$ is a deterministic function of $Y$.

We have explored two forms of pairwise potential:

$$\phi_{ij}(x_i, x_j) = \gamma_{x_i,x_j}[x_i \neq x_j], \quad \text{and} \quad \phi_{ij}(x_i, x_j) = (\sigma + \tau \, d_{ij})[x_i \neq x_j],$$

where $[\cdot]$ is one if its argument is true and zero otherwise, and $d_{ij}$ is some similarity measure over the appearance of the patches $i$ and $j$. In the first form, $\gamma_{x_i,x_j}$ is a general symmetric weight matrix that needs to be learned. The second potential is designed to favor label transitions at image locations with high contrast. As in [3] we use $d_{ij} = \exp(-\|z_i - z_j\|^2/(2\lambda))$, with $z_i \in \mathbb{R}^3$ denoting the average RGB value in the patch and $\lambda = \langle\|z_i - z_j\|^2\rangle$, the average L$_2$ norm between neighboring RGB values in the image. Models using the first form of potential will be denoted 'CRF$\gamma$' and those using the second will be denoted 'CRF$\tau$', or 'CRF$\sigma$' if $\tau$ has been fixed to zero. A graphical representation of the model is given in Figure 1.

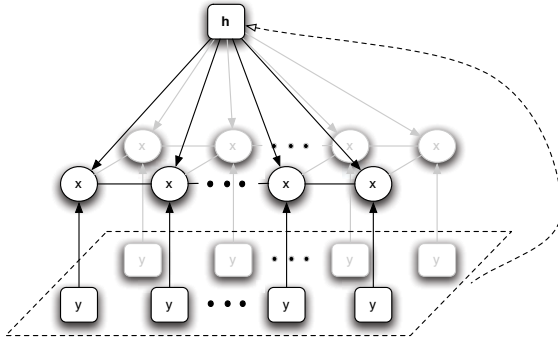

Figure 1: Graphical representation of the model with a single image-wide aggregate feature function denoted by $\mathbf{h}$. Squares denote feature functions and circles denote variable nodes $x_i$ (here connected in a 4-neighbor grid covering the image). Arrows denote single node potentials due to feature functions, and undirected edges represent pairwise potentials. The dashed lines indicate the aggregation of the single-patch observations $\mathbf{y}_i$ into $\mathbf{h}$.

## 3 Estimating a Conditional Random Field from Partially Labeled Images

Conditional models $p(X|Y)$ are usually trained by maximizing the log-likelihood of correct classification of the training data, $\sum_{n=1}^{N} \log p(X_n|Y_n)$. This requires completely labeled training data, i.e. a collection of $N$ pairs $(X_n, Y_n)_{n=1,...,N}$ with completely known $X_n$. In practice this is restrictive and it is useful to develop methods that can learn from partially labeled examples – images that include either completely unlabeled patches or ones with a retricted but nontrivial set of possible labels. Formally, we will assume that an incomplete labeling $X$ is known to belong to an associated set of admissible labelings $A$ and we maximise the log-likelihood for the model to predict any labeling in $A$:

$$
\begin{aligned}
L &= \log p(X \in A \,|\, Y) = \log \sum_{X \in A} p(X|Y) \\
&= \log \Big( \sum_{X \in A} \exp \big( - E(X|Y) \big) \Big) - \log \Big( \sum_{X} \exp \big( - E(X|Y) \big) \Big). \quad (4)
\end{aligned}
$$

Note that the log-likelihood is the difference between the partition functions of the restricted and unrestricted labelings, $p(X \,|\, Y, X \in A)$ and $p(X|Y)$. For completely labeled training images this reduces trivially to the standard labeled log-likelihood, while for partially labeled ones both terms of the log-likelihood are typically intractable because the set $A$ contains $O(C^k)$ distinct labelings $X$ where $k$ is the number of unlabeled patches and $C$ is the number of possible labels. Similarly, to find maximum likelihood parameter estimates using gradient descent we need to calculate partial derivatives with respect to each parameter $\theta$ and in general both terms are again intractable:

$$
\frac{\partial L}{\partial \theta} = \sum_{X} \Big( p(X|Y) - p(X \,|\, Y, X \in A) \Big) \frac{\partial E(X|Y)}{\partial \theta}. \quad (5)
$$

However the situation is not actually much worse than the fully-labeled case. In any case we need to approximate the full partition function $\log(\sum_X \exp -E(X|Y))$ or its derivatives and any method for doing so can also be applied to the more restricted sum $\log(\sum_{X \in A} \exp -E(X|Y))$ to give a contrast-of-partition-function based approximation. Here we will use the Bethe free energy approximation for both partition functions [14]:

$$
L \approx F_{Bethe}\big(p(X|Y)\big) - F_{Bethe}\big(p(X \,|\, Y, X \in A)\big). \quad (6)
$$

The Bethe approximation is a variational method based on approximating the complete distribution $p(X|Y)$ as the product of its pair-wise marginals (normalized by single-node marginals) that would apply if the graph were a tree. The necessary marginals are approximated using Loopy Belief Propagation (LBP) and the log-likelihood and its gradient are then evaluated using them [14]. Here LBP is run twice (with the singleton marginals initialized from the single node potentials), once to estimate the marginals of $p(X|Y)$ and once for $p(X \,|\, Y, X \in A)$. We used standard undamped LBP with uniform initial messages without encountering any convergence problems. In practice the approximate gradient and objective were consistent enough to allow parameter estimation using standard conjugate gradient optimization with adaptive step lengths based on monitoring the Bethe free-energy.

**Comparison with excision of unlabeled nodes.** The above training procedure requires two runs of loopy BP. A simple and often-used alternative is to discard unlabeled patches by excising nodes

| Class and frequency / Model | Building 16.1% | Grass 32.4% | Tree 12.3% | Cow 6.2% | Sky 15.4% | Plane 2.2% | Face 4.4% | Car 9.5% | Bike 1.5% | Per Pixel |
|---|---|---|---|---|---|---|---|---|---|---|
| IND loc only | 63.8 | 88.3 | 51.9 | 56.7 | 88.4 | 28.6 | 64.0 | 60.7 | 24.9 | 67.1 |
| IND loc+glo | 69.2 | 88.1 | 70.1 | 69.3 | 89.1 | 44.8 | 78.1 | 67.8 | 40.8 | 74.4 |
| CRF$\sigma$ loc only | 75.0 | 88.6 | 72.7 | 70.5 | 94.7 | 55.5 | 83.2 | 81.4 | 69.1 | 80.7 |
| CRF$\sigma$ loc+glo | 73.6 | 91.1 | 82.1 | 73.6 | 95.7 | 78.3 | 89.5 | 84.5 | 81.4 | **84.9** |
| CRF$\sigma$ loc+glo del unlabeled | 84.6 | 91.0 | 76.6 | 70.6 | 91.3 | 43.9 | 77.8 | 71.4 | 30.6 | 78.4 |
| CRF$\gamma$ loc only | 71.4 | 86.8 | 80.2 | 81.0 | 94.2 | 63.8 | 86.3 | 85.7 | 77.3 | 82.3 |
| CRF$\gamma$ loc+glo | 74.6 | 88.7 | 82.5 | 82.2 | 93.9 | 61.7 | 88.8 | 82.8 | 76.8 | 83.3 |
| CRF$\tau$ loc only | 65.6 | 85.4 | 78.2 | 74.3 | 95.4 | 61.8 | 84.8 | 85.2 | 79.4 | 80.3 |
| CRF$\tau$ loc+glo | 75.0 | 88.5 | 82.3 | 81.0 | 94.4 | 60.6 | 88.7 | 82.2 | 76.1 | 83.1 |
| Schroff et al. [15] | 56.7 | 84.8 | 76.4 | 83.8 | 81.1 | 53.8 | 68.5 | 71.4 | 72.0 | 75.2 |
| PLSA-MRF [10] | 74.0 | 88.7 | 64.4 | 77.4 | 95.7 | 92.2 | 88.8 | 81.1 | 78.7 | 82.3 |

Table 1: Classification accuracies on the 9 MSRC classes using different models. For each class its frequency in the ground truth labeling is also given.

that correspond to unlabeled or partially labeled patches from the graph. This leaves a random field with one or more completely labeled connected components whose log-likelihood $p(X'|Y')$ we maximize directly using gradient based methods. Equivalently, we can use the complete model but set all of the pair-wise potentials connected to unlabeled nodes to zero: this decouples the labels of the unlabeled nodes from the rest of the field. As a result $p(X|Y)$ and $p(X\,|\,Y, X \in A)$ are equivalent for the unlabeled nodes and their contribution to the log-likelihood in Eq. (4) and the gradient in Eq. (5) vanishes.

The problem with this approach is that it systematically overestimates spatial coupling strengths. Looking at the training labelings in Figure 3 and Figure 4, we see that pixels near class boundaries often remain unlabeled. Since we leave patches unlabeled if they contain unlabeled pixels, label transitions are underrepresented in the training data, which causes the strength of the pairwise couplings to be greatly overestimated. In contrast, the full CRF model provides realistic estimates because it is forced to include a (fully coupled) label transition somewhere in the unlabeled region.

## 4 Experimental Results

This section analyzes the performance of our segmentation models in detail and compares it to other existing methods. In our first set of experiments we use the Microsoft Research Cambridge (MSRC) dataset[1]. This consists of 240 images of $213 \times 320$ pixels and their partial pixel-level labelings. The labelings assign pixels to one of nine classes: *building, grass, tree, cow, sky, plane, face, car,* and *bike.* About 30% of the pixels are unlabeled. Some sample images and labelings are shown in Figure 4. In our experiments we divide the dataset into 120 images for training and 120 for testing, reporting average results over 20 random train-test partitions. We used $20 \times 20$ pixel patches with centers at 10 pixel intervals. (For the patch size see the red disc in Figure 4).

To obtain a labeling of the patches, pixels are assigned to the nearest patch center. Patches are allowed to have any label seen among their pixels, with unlabeled pixels being allowed to have any label. Learning and inference takes place at the patch level. To map the patch-level segmentation back to the pixel level we assign each pixel the marginal of the patch with the nearest center. (In Figure 4 the segmentations were post-processed by a applying a Gaussian filter over the pixel marginals with the scale set to half the patch spacing). The performance metrics ignore unlabeled test pixels.

The relative contributions of the different components of our model are summarized in Table 1. Models that incorporate 4-neighbor spatial couplings are denoted 'CRF' while ones that incorporate only (local or global) patch-level potentials are denoted 'IND'. Models that include global aggregate features are denoted 'loc+glo', while ones that include only on local patch-level features are denoted 'loc only'.

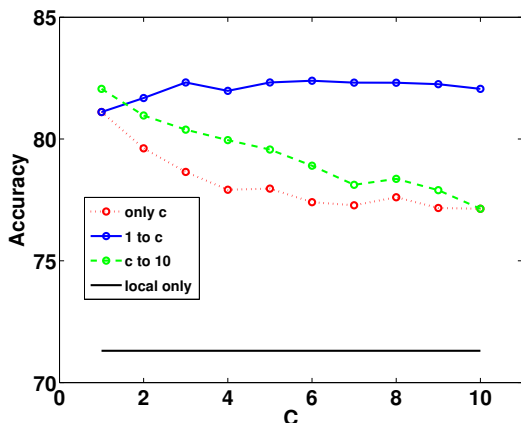

Figure 2: Classification accuracy as a function of the aggregation fineness $c$, for the 'IND' (individual patch) classifier using a single training and test set. Aggregate features (AF) were computed in each cell of a $c \times c$ image partition. Results are given for models with no AFs (solid line), with AFs of a single $c$ (dotted curve), with AFs on grids $1 \times 1$ up to $c \times c$ (solid curve), and with AFs on grids $c \times c$ up to $10 \times 10$ (dashed curve).

**Benefits of aggregate features.** The first main conclusion is that including global aggregate features helps, for example improving the average classification rate on the MSRC dataset from 67.1% to 74.4% for the spatially uncoupled 'IND' model and from 80.7% to 84.9% for the 'CRF$\sigma$' spatial model.

The idea of aggregation can be generalized to scales smaller than the complete image. We experimented with dividing the image into $c \times c$ grids for a range of values of $c$. In each cell of the grid we compute a separate histogram over the visual words, and for each patch in the cell we include an energy term based on this histogram in the same way as for the image-wide histogram in Eq. (1). Figure 2 shows how the performance of the individual patch classifier depends on the use of aggregate features. From the dotted curve in the figure we see that although using larger cells to aggregate features is generally more informative, even fine $10 \times 10$ subdivisions (containing only 6–12 patches per cell) provide a significant performance increase. Furthermore, including aggregates computed at several different scales does help, but the performance increment is small compared to the gain obtained with just image-wide aggregates. Therefore we included only image-wide aggregates in the subsequent experiments.

**Benefits of including spatial coupling.** The second main conclusion from Table 1 is that including spatial couplings (pairwise CRF potentials) helps, respectively increasing the accuracy by 10.5% for 'loc+glo' and by 13.6% for 'loc only' for 'CRF$\sigma$' relative to 'IND'. The improvement is particularly noticeable for rare classes when global aggregate features are not included: in this case the single node potentials are less informative and frequent classes tend to be unduly favored due to their large a priori probability.

When the image-wide aggregate features are included ('loc+glo'), the simplest pairwise potential – the 'CRF$\sigma$' Potts model – works better than the more general models 'CRF$\gamma$' and 'CRF$\tau$', while if only the local features are included ('loc only'), the class-dependent pairwise potential 'CRF$\gamma$' works best. The performance increment from global features is smallest for 'CRF$\gamma$', the model that also includes local contextual information. The overall influence of the local label transition preferences expressed in 'CRF$\gamma$' appears to be similar to that of the global contextual information provided by image-wide aggregate features.

**Benefits of training by marginalizing partial labelings.** Our third main conclusion from Table 1 is that our marginalization based training method for handling missing labels is superior to the common heuristic of deleting any unlabeled patches. Learning a 'CRF$\sigma$ loc+glo' model by removing all unlabeled patches ('del unlabeled' in the table) leads to an estimate $\sigma \approx 11.5$, whereas the maximum likelihood estimate of (4) leads to $\sigma \approx 1.9$. In particular, with 'delete unlabeled' training the accuracy of the model drops significantly for the classes *plane* and *bike*, both of which have a relatively small area relative to their boundaries and thus many partially labeled patches. It is interesting to note that even though $\sigma$ has been severely over-estimated in the 'delete unlabeled' model, the CRF still improves over the individual patch classification obtained with 'IND loc+glo' for most classes, albeit not for *bike* and only marginally for *plane*.

**Recognition as function of the amount of labeling.** We now consider how the performance drops as the fraction of labeled pixels decreases. We applied a morphological erosion operator to the manual annotations, where we varied the size of the disk-shaped structuring element from $0, 5, \ldots, 50$.

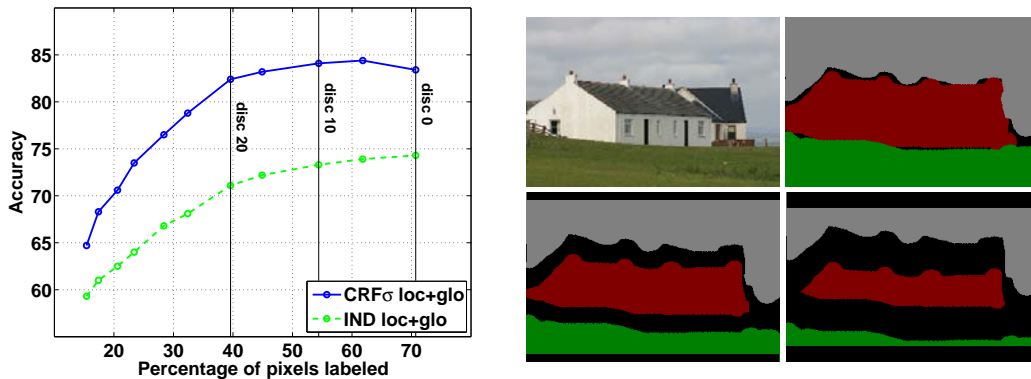

Figure 3: Recognition performance when learning from increasingly eroded label images (left). Example image with its original annotation, and erosions thereof with disk of size 10 and 20 (right).

In this way we obtain a series of annotations that resemble increasingly sloppy manual annotations, see Figure 3. The figure also shows the recognition performance of 'CRF$\sigma$ loc+glo' and 'IND loc+glo' as a function of the fraction of labeled pixels. In addition to its superior performance when trained on well labeled images, the CRF maintains its performance better as the labelling becomes sparser. Note that 'CRF$\sigma$ loc+glo' learned from label images eroded with a disc of radius 30 (only 28% of pixels labeled) still outperforms 'IND loc+glo' learned from the original labeling (71% of pixels labeled). Also, the CRF actually performs better with 5 pixels of erosion than with the original labeling, presumably because ambiguities related to training patches with mixed pixel labels are reduced.

**Comparison with related work.** Table 1 also compares our recognition results on the MSRC dataset with those reported in [15, 10]. Our CRF model clearly outperforms the approach of [15], which uses aggregate features of an optimized scale but lacks spatial coupling in a random field, giving a performance very similar to that of our 'IND loc+glo' model. Our CRF model also performs slightly better than our generative approach of [10], which is based on the same feature set but differs in its implementation of image-wide contextual information ([10] also used a 90%–10% training-test partition, not 50%-50% as here).

Using the Sowerby dataset and a subset of the Corel dataset we also compare our model with two CRF models that operate at pixel-level. The Sowerby dataset consists of 104 images of $96 \times 64$ pixels of urban and rural scenes labeled with 7 different classes: *sky, vegetation, road marking, road surface, building, street objects* and *cars*. The subset of the Corel dataset contains 100 images of $180 \times 120$ pixels of natural scenes, also labeled with 7 classes: *rhino/hippo, polar bear, water, snow, vegetation, ground,* and *sky*. Here we used $10 \times 10$ pixel patches, with a spacing of respectively 2 and 5 pixels for the Sowerby and Corel datasets. The other parameters were kept as before. Table 2 compares the recognition accuracies averaged over pixels for our CRF and independent patch models to the results reported on these datasets for TextonBoost [7] and the multi-scale CRF model of [5]. In this table 'IND' stands for results obtained when only the single node potentials are used in the respective models, disregarding the spatial random field couplings. The total training time and test time per image are listed for the full CRF models. The results show that on these datasets our model performs comparably to pixel-level approaches while being much faster to train and test since it operates at patch-level and uses standard features as opposed to the boosting procedure of [7].

## 5 Conclusion

We presented several image-patch-level CRF models for semantic image labeling that incorporate both local patch-level observations and more global contextual features based on aggregates of observations at several scales. We showed that partially labeled training images could be handled by maximizing the total likelihood of the image segmentations that comply with the partial labeling, using Loopy BP and Bethe free-energy approximations for the calculations. This allowed us to learn effective CRF models from images where only a small fraction of the pixels were labeled and class transitions were not observed. Experiments on the MSRC dataset showed that including image-

| | Sowerby | | | | Corel | | | |
|---|---|---|---|---|---|---|---|---|
| | Accuracy | | Speed | | Accuracy | | Speed | |
| | IND | CRF | train | test | IND | CRF | train | test |
| TextonBoost [7] | 85.6% | 88.6% | 5h | 10s | 68.4% | 74.6% | 12h | 30s |
| He et al. [5] CRF | 82.4% | 89.5% | Gibbs | Gibbs | 66.9% | 80.0% | Gibbs | Gibbs |
| CRF$\sigma$ loc+glo | 86.0% | 87.4% | 20min | 5s | 66.9% | 74.6 % | 15min | 3s |

Table 2: Recognition accuracy and speeds on the Corel and Sowerby dataset.

wide aggregate features is very helpful, while including additional aggregates at finer scales gives relatively little further improvement. Comparative experiments showed that our patch-level CRFs have comparable performance to state-of-the-art pixel-level models while being much more efficient because the number of patches is much smaller than the number of pixels.

## Footnotes

[1]Available from `http://research.microsoft.com/vision/cambridge/recognition`.

# References

[1] P. Jansen, W. van der Mark, W. van den Heuvel, and F. Groen. Colour based off-road environment and terrain type classification. In *Proceedings of the IEEE Conference on Intelligent Transportation Systems*, pages 216–221, 2005.

[2] S. Geman and D. Geman. Stochastic relaxation, Gibbs distributions and the Bayesian restoration of images. *IEEE Transactions on Pattern Analysis and Machine Intelligence*, 6(6):712–741, 1984.

[3] C. Rother, V. Kolmogorov, and A. Blake. GrabCut: interactive foreground extraction using iterated graph cuts. *ACM Transactions on Graphics*, 23(3):309–314, 2004.

[4] J. Lafferty, A. McCallum, and F. Pereira. Conditional random fields: probabilistic models for segmenting and labeling sequence data. In *Proceedings of the International Conference on Machine Learning*, volume 18, pages 282–289, 2001.

[5] X. He, R. Zemel, and M. Carreira-Perpiñán. Multiscale conditional random fields for image labelling. In *Proceedings of the IEEE Conference on Computer Vision and Pattern Recognition*, pages 695–702, 2004.

[6] S. Kumar and M. Hebert. A hierarchical field framework for unified context-based classification. In *Proceedings of the IEEE International Conference on Computer Vision*, pages 1284–1291, 2005.

[7] J. Shotton, J. Winn, C. Rother, and A. Criminisi. Textonboost: joint appearance, shape and context modeling for multi-class object recognition and segmentation. In *Proceedings of the European Conference on Computer Vision*, pages 1–15, 2006.

[8] A. Quattoni, M. Collins, and T. Darrell. Conditional random fields for object recognition. In *Advances in Neural Information Processing Systems*, volume 17, pages 1097–1104, 2005.

[9] P. Carbonetto, G. Dorkó, C. Schmid, H. Kück, and N. de Freitas. A semi-supervised learning approach to object recognition with spatial integration of local features and segmentation cues. In *Toward Category-Level Object Recognition*, pages 277–300, 2006.

[10] J. Verbeek and B. Triggs. Region classification with Markov field aspect models. In *Proceedings of the IEEE Conference on Computer Vision and Pattern Recognition*, 2007.

[11] D. Lowe. Distinctive image features from scale-invariant keypoints. *International Journal of Computer Vision*, 60(2):91–110, 2004.

[12] J. van de Weijer and C. Schmid. Coloring local feature extraction. In *Proceedings of the European Conference on Computer Vision*, pages 334–348, 2006.

[13] The 2005 PASCAL visual object classes challenge. In F. d'Alche-Buc, I. Dagan, and J. Quinonero, editors, *Machine Learning Challenges: Evaluating Predictive Uncertainty, Visual Object Classification, and Recognizing Textual Entailment, First PASCAL Machine Learning Challenges Workshop*. Springer, 2006.

[14] J. Yedidia, W. Freeman, and Y. Weiss. Understanding belief propagation and its generalizations. Technical Report TR-2001-22, Mitsubishi Electric Research Laboratories, 2001.

[15] F. Schroff, A. Criminisi, and A. Zisserman. Single-histogram class models for image segmentation. In *Proceedings of the Indian Conference on Computer Vision, Graphics and Image Processing*, 2006.

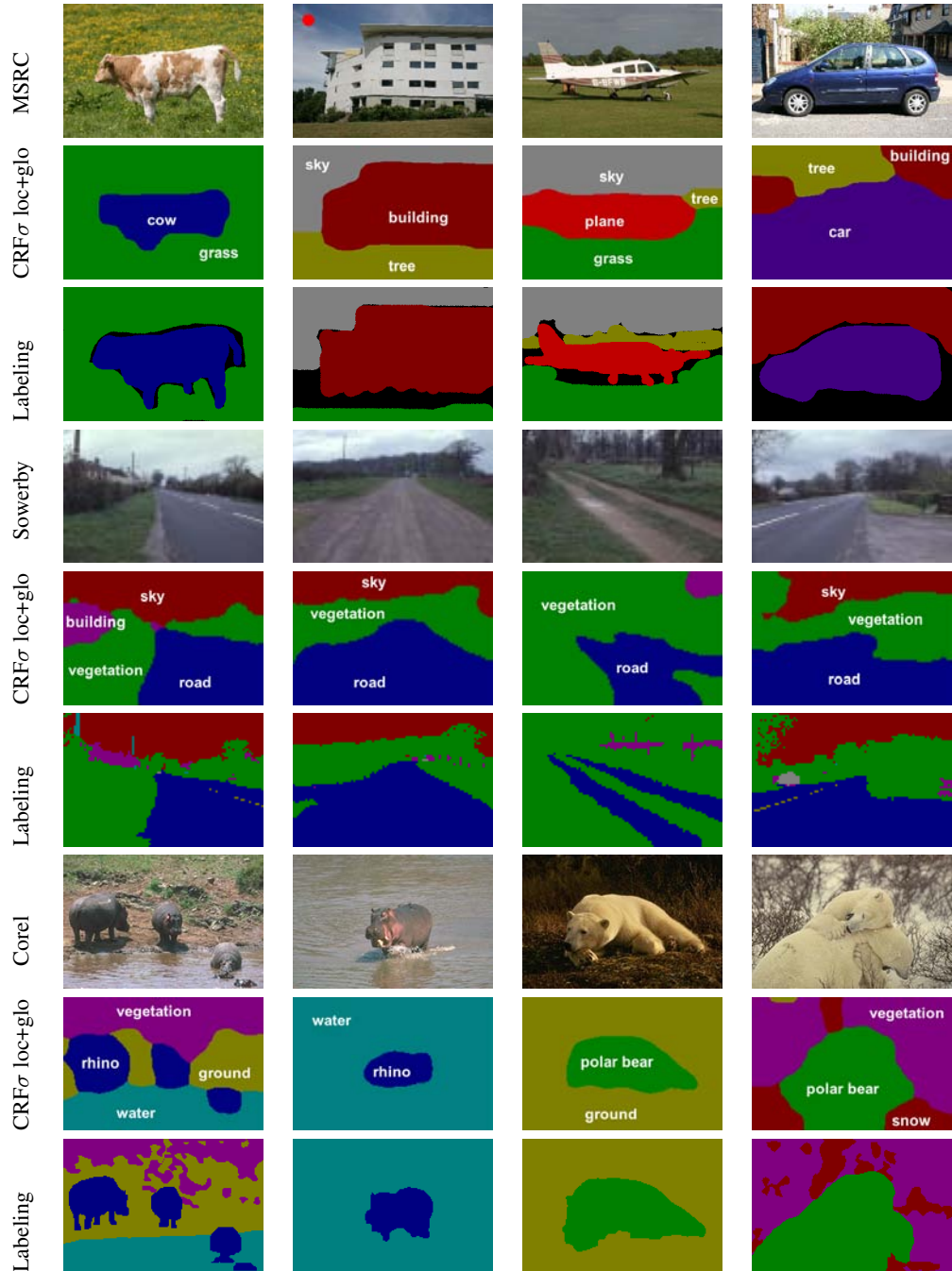

Figure 4: Samples from the MSRC, Sowerby, and Corel datasets with segmentation and labeling.

